# Learning Instance-Independent Value Functions to Enhance Local Search

**Robert Moll    Andrew G. Barto    Theodore J. Perkins**
Department of Computer Science
University of Massachusetts, Amherst, MA 01003

**Richard S. Sutton**
AT&T Shannon Laboratory, 180 Park Avenue, Florham Park, NJ 07932

## Abstract

Reinforcement learning methods can be used to improve the performance of local search algorithms for combinatorial optimization by learning an evaluation function that predicts the outcome of search. The evaluation function is therefore able to guide search to low-cost solutions better than can the original cost function. We describe a reinforcement learning method for enhancing local search that combines aspects of previous work by Zhang and Dietterich (1995) and Boyan and Moore (1997, Boyan 1998). In an off-line learning phase, a value function is learned that is useful for guiding search for multiple problem sizes and instances. We illustrate our technique by developing several such functions for the Dial-A-Ride Problem. Our learning-enhanced local search algorithm exhibits an improvement of more then 30% over a standard local search algorithm.

## 1  INTRODUCTION

Combinatorial optimization is of great importance in computer science, engineering, and operations research. We investigated the use of reinforcement learning (RL) to enhance traditional local search optimization (hillclimbing). Since local search is a sequential decision process. RL can be used to improve search performance by learning an evaluation function that predicts the outcome of search and is therefore able to guide search to low-cost solutions better than can the original cost function.

Three approaches to using RL to improve combinatorial optimization have been described

in the literature. One is to learn a value function over multiple search trajectories of a single problem instance. As the value function improves in its predictive accuracy, its guidance enhances additional search trajectories on the same instance. Boyan and Moore's STAGE algorithm (Boyan and Moore 1997, Boyan 1998) falls into this category, showing excellent performance on a range of optimization problems. Another approach is to learn a value function off-line and then use it over multiple new instances of the same problem. Zhang and Dietterich's (1995) application of RL to a NASA space shuttle mission scheduling problem takes this approach (although it does not strictly involve local search as we define it below). A key issue here is the need to normalize state representations and rewards so that trajectories from instances of different sizes and difficulties yield consistent training data. In each of the above approaches, a state of the RL problem is an entire solution (e.g., a complete tour in a Traveling Salesman Problem (TSP)) and the actions select next solutions from the current solutions' neighborhoods. A third approach, described by Bertsekas and Tsitsiklis (1996), uses a learned value function for guiding the direct construction of solutions rather than for moving between them.

We focused on combining aspects of first two of these approaches with the goal of carefully examining how well the TD($\lambda$) algorithm can learn an instance-independent value function for a given problem to produce an enhanced local search algorithm applicable to all instances of that problem. Our approach combines an off-line learning phase with STAGE's alternation between using the learned value function and the original cost function to guide search. We present an extended case study of this algorithm's application to a somewhat complicated variant of TSP known as the Dial-A-Ride Problem, which exhibits some of the non-uniform structure present in real-world transportation and logistics problems.

## 2  ENHANCING LOCAL SEARCH

The components of local search for combinatorial optimization are 1) a finite set of *feasible solutions*, $S$; 2) an *objective, or cost, function*, $c : S \to \Re$; and 3) a *neighborhood function*, $A : S \to \mathcal{P}(\mathcal{S})$ (the power set of $S$). Local search starts with an initial feasible solution, $s_0$, of a problem instance and then at each step $k = 1, 2, \ldots$, it selects a solution $s_k \in A(s_{k-1})$ such that $c(s_k) < c(s_{k-1})$. This process continues until further local improvement is impossible, and the current *local optimum* is returned. If the algorithm always moves to the *first* less expensive neighboring solution encountered in an enumeration of a neighborhood, it is called *first improvement* local search.

Following Zhang and Dietterich (1995) and Boyan and Moore (1997), we note that local search can be viewed as a policy of a Markov decision process (MDP) with state set $S$ and action sets $A(s)$, $s \in S$, where an action is identified with the neighboring solution selected. Local search selects actions which decrease the value of $c$, eventually absorbing at a state with a locally minimum cost. But $c$ is not the optimal value function for the local search problem, whose objective is to reach the lowest-cost absorbing state (possibly including some tradeoff involving the number of search steps required to do so). RL used with a function approximator can learn an approximate optimal value function, $V$, thereby producing an enhanced search algorithm that is locally guided by $V$ instead of by $c$. One way to do this is to give a small penalty, $\epsilon$, for each transition and a terminal reward upon absorption that is inversely related to the cost of the terminal state. Maximizing the expected undiscounted return accomplishes the desired tradeoff (determined by the value of $\epsilon$) between quality of final solution and search time (cf. Zhang and Dietterich, 1995).

Since each instance of an optimization problem corresponds to a different MDP, a value

function $V$ learned in this way is instance-specific. Whereas Boyan's STAGE algorithm in effect uses such a $V$ to enhance additional searches that start from different states of the same instance, we are interested in learning a $V$ off-line, and then using it for arbitrary instances of the given problem. In this case, the relevant sequential decision problem is more complicated than a single-instance MDP since it is a summary of aspects of all problem instances. It would be extremely difficult to make the structure of this process explicit, but fortunately RL requires only the generation of sample trajectories, which is relatively easy in this case.

In addition to their cost, secondary characteristics of feasible solutions can provide valuable information for search algorithms. By adjusting the parameters of a function approximation system whose inputs are feature vectors describing feasible solutions, an RL algorithm can produce a compact representation of $V$. Our approach operates in two distinct phases. In the *learning phase*, it learns a value function by applying the TD($\lambda$) algorithm to a number of randomly chosen instances of the problem. In the *performance phase*, it uses the resulting value function, now held fixed, to guide local search for additional problem instances. This approach is in principle applicable to any combinatorial optimization problem, but we describe its details in the context of the Dial-A-Ride problem.

## 3 THE DIAL-A-RIDE PROBLEM

The Dial-a-Ride Problem (DARP) has the following formulation. A van is parked at a terminal. The driver receives calls from $N$ customers who need rides. Each call identifies the location of a customer, as well as that customer's destination. After the calls have been received, the van must be routed so that it starts from the terminal, visits each pick-up and drop-off site in some order, and then returns to the terminal. The tour must pick up a passenger before eventually dropping that passenger off. The tour should be of minimal length. Failing this goal—and DARP is NP-complete, so it is unlikely that optimal DARP tours will be found easily—at least a good quality tour should be constructed. We assume that the van has unlimited capacity and that the distances between pick-up and drop-off locations are represented by a symmetric Euclidean distance matrix.

We use the notation

$$0 \quad 1 \quad 2 \quad -1 \quad 3 \quad -3 \quad -2$$

to denote the following tour: "start at the terminal (0), then pick up 1, then 2, then drop off 1 (thus: $-1$), pick up 3, drop off 3, drop off 2 and then return to the terminal (site 0)." Given a tour $s$, the *2-opt neighborhood* of $s$, $A_2(s)$, is the set of legal tours obtainable from $s$ by subsequence reversal. For example, for the tour above, the new tour created by the following subsequence reversal

$$0 \ 1 \ / \ 2 \ -1 \ 3 \ / \ -3 \ -2 \ \longrightarrow \ 0 \ 1 \ 3 \ -1 \ 2 \ -3 \ -2$$

is an element of $A_2(T)$. However, this reversal

$$0 \ 1 \ 2 \ / \ -1 \ 3 \ -3 \ / \ -2 \ \longrightarrow \ 0 \ 1 \ 2 \ -3 \ 3 \ -1 \ -2$$

leads to an infeasible tour, since it asserts that passenger 3 is dropped off first, then picked up. The neighborhood structure of DARP is highly non-uniform, varying between $A_2$ neighborhood sizes of $O(N)$ and $O(N^2)$.

Let $s$ be a feasible DARP tour. By 2-opt($s$) we mean the tour obtained by first-improvement local search using the $A_2$ neighborhood structure (presented in a fixed, standard enumeration), with tour length as the cost function. As with TSP, there is a 3-opt algorithm for

DARP, where a 3-opt neighborhood $A_3(s)$ is defined and searched in a fixed, systematic way, again in first-improvement style. This neighborhood is created by inserting three rather than two "breaks" in a tour. 3-opt is much slower than 2-opt, more than 100 times as slow for $N = 50$, but it is much more effective, even when 2-opt is given equal time to generate multiple random starting tours and then complete its improvement scheme.

Psaraftis (1983) was the first to study 2-opt and 3-opt algorithms for DARP. He studied tours up to size $N = 30$, reporting that at that size, 3-opt tours are about 30% shorter on average than 2-opt tours. In theoretical studies of DARP, Stein (1978) showed that for sites placed in the unit square, the globally optimal tour for problem size $N$ has a length that asymptotically approaches $1.02\sqrt{2N}$ with probability 1 as $N$ increases. This bound applies to our study—although we multiply position coordinates by 100 and then truncate to get integer distance matrices—and thus, for example, a value of 1020 gives us a baseline estimate of the globally optimal tour cost for $N = 50$. Healy and Moll (1995) considered using a secondary cost function to extend local search on DARP. In addition to primary cost (tour length) they considered as a secondary cost the ratio of tour cost to neighborhood size, which they called *cost-hood*. Their algorithm employed a STAGE-like alternation between these two cost functions: starting from a random tour $s$, it first found 2opt($s$); then it performed a limited local search using the cost-hood function, which had the effect of driving the search to a new tour with a decent cost and a large neighborhood. These alternating processes were repeated until a time bound was exhausted, at which point the least cost tour seen so far was reported as the result of the search. This technique worked well, with effectiveness falling midway between that of 2-opt and 3-opt.

## 4 ENHANCED 2-OPT FOR DARP

We restrict our description to a learning method for enhancing 2-opt for DARP, but the same method can be used for other problems. In the learning phase, after initializing the function approximator, we conduct a number training episodes until we are satisfied that the weights have stabilized. For each episode $k$, we select a problem size $N$ at random (from a predetermined range) and generate a random DARP instance of that size, i.e., we generate a symmetric Euclidean distance matrix by generating random points in the plane inside the square bounded by the points (0,0), (0,100), (100,100) and (100,0). We set the "terminal site" to point (50,50) and the initial tour to a randomly generated feasible tour. We then conduct a modified first-improvement 2-opt local search using the negated current value function, $-V_k$, as the cost function. The modification is that termination is controlled by a parameter $\epsilon > 0$ as follows: the search terminates at a tour $s$ if there is no $s' \in A(s)$ such that $V_k(s') > V_k(s) + \epsilon$. In other words, a step is taken only if it produces an improvement of at least $\epsilon$ according to the current value function. The episode returns a final tour $s_f$. We run one unmodified 2-opt local search, this time using the DARP cost function $c$ (tour length), from $s_f$ to compute 2-opt($s_f$). We then apply a batch version of undiscounted TD($\lambda$) to the saved search trajectory using the following immediate rewards: $-\epsilon$ for each transition, and $-c(\text{2-opt}(s_f))/Stein_N$ as a terminal reward, where $Stein_N$ is the Stein estimate for instance size $N$. Normalization by $Stein_N$ helps make the terminal reward consistent across instance sizes. At the end of this learning phase, we have a final value function, $V$. $V$ is used in the performance phase, which consists of applying the modified first-improvement 2-opt local search with cost function $-V$ on new instances, followed by a 2-opt application to the resulting tour.

The results described here were obtained using a simple linear approximator with a bias

Table 1: Weight Vectors for Learned Value Functions.

| Value Function | Weight Vector |
|---|---|
| $V$ | $< .951, .033, .0153 >$ |
| $V_{20}$ | $< .981, .019, .00017 >$ |
| $V_{30}$ | $< .984, .014, .0006 >$ |
| $V_{40}$ | $< .977, .022, .0009 >$ |
| $V_{50}$ | $< .980, .019, .0015 >$ |
| $V_{60}$ | $< .971, .022, .0069 >$ |

weight and features developed from the following base features: 1) $normcost_N(s) = c(s)/Stein_N$; 2) $normhood_N = |A(s)|/a_N$, where $a_N$ is a normalization coefficient defined below; and 3) $normprox_N$, which considers a list of the $N/4$ least expensive edges of the distance matrix, as follows. Let $e$ be one of the edges, with endpoints $u$ and $v$. The $normprox_N$ feature examines the current tour, and counts the number of sites on the tour that appear between $u$ and $v$. $normprox_N$ is the sum of these counts over the edges on the proximity list divided by a normalizing coefficient $b_N$ described below. Our function approximator is then give by $w_0 + normcost_N/(normhood_N)^2 w_1 + normprox_N/(normhood_N)^2 w_2$. The coefficients $a_N$ and $b_N$ are the result of running linear regression on randomly sampled instances of random sizes to determine coefficients that will yield the closest fit to a constant target value for normalized neighborhood size and proximity. The results were $a_N = .383N^2 + .28.5N - 244.5$ and $b_N = .43N^2 + .736N - 68.9\sqrt{N} + 181.75$. The motivation for the quotient features comes from Healy and Moll (1995) who found that using a similar term improved 2-opt on DARP by allowing it to sacrifice cost improvements to gain large neighborhoods.

## 5 EXPERIMENTAL RESULTS

Comparisons among algorithms were done at five representative sizes $N = 20, 30, 40, 50,$ and 60. For the learning phase, we conducted approximately 3,000 learning episodes, each one using a randomly generated instance of size selected randomly between 20 and 60 inclusive. The result of the learning phase was a value function $V$. To assess the influence of this multi-instance learning, we also repeated the above learning phase 5 times, except that in each we held the instance size fixed to a different one of the 5 representative sizes, yielding in each case a distinct value function $V_N$, where $N$ is the training instance size. Table 1 shows the resulting weight vector $<$ bias weight, $costhood_N$ weight, $proximity_N$ weight $>$. With the exception of the $proximity_N$ weight, these are quite consistent across training instance size. We do not yet understand why training on multiple-sized instances led to this pattern of variation.

Table 2 compares the tour quality found by six different local search algorithms. For the algorithms using learned value functions, the results are for the performance phase after learning using the algorithm listed. Table entries are the percent by which tour length exceeded $Stein_N$ for instance size $N$ averaged over 100 instances of each representative size. Thus, 2-opt exceeded $Stein_{20} = 645$ on the 100 instance sample set by an average of 42%. The last row in the table gives the results of using the five different value functions $V_N$, for the corresponding $N$. Results for TD(.8) are shown because they were better than

Table 2: Comparison of Six Algorithms at Sizes $N$ = 20, 30, 40, 50, 60. Entries are percentage above $Stein_N$ averaged over 100 random instances of size $N$.

| Algorithm | N=20 | N=30 | N=40 | N=50 | N=60 |
|---|---|---|---|---|---|
| 2-opt | 42 | 47 | 53 | 56 | 60 |
| 3-opt | 8 | 8 | 11 | 10 | 10 |
| TD(1) | 28 | 31 | 34 | 39 | 40 |
| TD(.8) $\epsilon = 0$ | 27 | 30 | 35 | 37 | 39 |
| TD(.8) $\epsilon = .01/N$ | 29 | 35 | 37 | 41 | 44 |
| TD(.8) $\epsilon = 0$, $V_N$ | 29 | 30 | 32 | 36 | 40 |

Table 3: Average Relative Running Times. Times for 2-opt are in seconds; other entries give time divided by 2-opt time.

| Algorithm | N=20 | N=30 | N=40 | N=50 | N=60 |
|---|---|---|---|---|---|
| 2-opt | .237 | .770 | 1.09 | 1.95 | 3.55 |
| 3-opt | 32 | 45 | 100 | 162 | 238 |
| TD(.8) $\epsilon = 0$ | 3.2 | 3.4 | 6.3 | 6.9 | 7.1 |
| TD(.8) $\epsilon = .01/N$ | 2.2 | 1.8 | 2.6 | 2.9 | 3.0 |

those for other values of $\lambda$. The learning-enhanced algorithms do well against 2-opt when running time is ignored, and indeed TD(.8), $\epsilon = 0$, is about 35% percent better (according to this measure) by size 60. Note that 3-opt clearly produces the best tours, and a non-zero $\epsilon$ for TD(.8) decreases tour quality, as expected since it causes shorter search trajectories.

Table 3 gives the relative running times of the various algorithms. The raw running times for 2-opt are given in seconds (Common Lisp on 266 Mhz Mac G-3) at each of five sizes in the first row. Subsequent rows give approximate running times divided by the corresponding 2-opt running time. Times are averages over 30 instances. The algorithms using learned value functions are slower mainly due to the necessity to evaluate the features. Note that TD(.8) becomes significantly faster with $\epsilon$ non-zero.

Finally, Table 4 gives the relative performance of seven algorithms, normalized for time, including the STAGE algorithm using linear regression with our features. We generated 20 random instances at each of the representative sizes, and we allowed each algorithm to run for the indicated amount of time on each instance. If time remained when a local optimum was reached, we restarted the algorithm at that point, except in the case of 2-opt, where we selected a new random starting tour. The restarting regime for the learning-enhanced algorithms is the regime employed by STAGE. Each algorithm reports the best result found in the allotted time, and the chart reports the averages of these values across the 20 instances. Notice that the algorithms that take advantage of extensive off-line learning significantly outperform the other algorithms, including STAGE, which relies on single-instance learning.

## 6  DISCUSSION

We have presented an extension to local search that uses RL to enhance the local search cost function for a particular optimization problem. Our method combines aspects of work

Table 4: Performance Comparisons, Equalized for Running Time.

| | Size and Running Time | | | | |
|---|---|---|---|---|---|
| | N=20 | N=30 | N=40 | N=50 | N=60 |
| Algorithm | 10 sec | 20 sec | 40 sec | 100 sec | 150 sec |
| 2-opt | 16 | 29 | 28 | 30 | 38 |
| STAGE | 18 | 20 | 32 | 24 | 27 |
| TD(.8) $\epsilon = 0$ | 12 | 13 | 16 | 22 | 20 |
| TD(.8) $\epsilon = .01/N$ | 13 | 11 | 14 | 24 | 28 |

by Zhang and Dietterich (1995) and Boyan and Moore (1997; Boyan 1998). We have applied our method to a relatively pure optimization problem—DARP—which possesses a relatively consistent structure across problem instances. This has allowed the method to learn a value function that can be applied across all problem instances at all sizes. Our method yields significant improvement over a traditional local search approach to DARP on the basis of a very simple linear approximator, built using a relatively impoverished set of features. It also improves upon Boyan and Moore's (1997) STAGE algorithm in our example problem, benefiting from extensive off-line learning whose cost was not included in our assessment. We think this is appropriate for some types of problems; since it is a one-time learning cost, it can be amortized over many future problem instances of practical importance.

## Acknowledgement

We thank Justin Boyan for very helpful discussions of this subject. This research was supported by a grant from the Air Force Office of Scientific Research, Bolling AFB (AFOSR F49620-96-1-0254).

## References

Boyan, J. A. (1998). Learning Evaluation Functions for Global Optimization. Ph.D. Thesis, Carnegie-Mellon University.

Boyan, J. A., and Moore, A. W. (1997). Using Prediction to Improve Combinatorial Optimization Search. *Proceedings of AI-STATS-97*.

D. P. Bertsekas, D. P., and Tsitsiklis, J. N. (1996). *Neuro-Dynamic Programming*. Athena Scientific, Belmont, MA.

Healy, P., and Moll, R. (1995). A New Extension to Local Search Applied to the Dial-A-Ride Problem. *European Journal of Operations Research*, 8: 83–104.

Psaraftis, H. N. (1983). K-interchange Procedures for Local Search in a Precedence-Constrained Routing Problem. *European Journal of Operations Research*, 13:391–402.

Zhang, W. and Dietterich, T. G. (1995). A Reinforcement Learning Approach to Job-Shop Scheduling. In *Proceedings of the Fourteenth International Joint Conference on Artificial Intelligence*, pp. 1114–1120. Morgan Kaufmann, San Francisco.

Stein, D. M. (1978). An Asymptotic Probabilistic Analysis of a Routing Problem. *Math. Operations Res. J.*, 3: 89–101.